# Common-Frame Model for Object Recognition

**Pierre Moreels**          **Pietro Perona**
California Insitute of Technology - Pasadena CA91125 - USA
`pmoreels,perona@vision.caltech.edu`

## Abstract

A generative probabilistic model for objects in images is presented. An object consists of a constellation of features. Feature appearance and pose are modeled probabilistically. Scene images are generated by drawing a set of objects from a given database, with random clutter sprinkled on the remaining image surface. Occlusion is allowed.

We study the case where features from the same object share a common reference frame. Moreover, parameters for shape and appearance densities are shared across features. This is to be contrasted with previous work on probabilistic 'constellation' models where features depend on each other, and each feature and model have different pose and appearance statistics [1, 2]. These two differences allow us to build models containing hundreds of features, as well as to train each model from a single example. Our model may also be thought of as a probabilistic revisitation of Lowe's model [3, 4].

We propose an efficient entropy-minimization inference algorithm that constructs the best interpretation of a scene as a collection of objects and clutter. We test our ideas with experiments on two image databases. We compare with Lowe's algorithm and demonstrate better performance, in particular in presence of large amounts of background clutter.

## 1 Introduction

There is broad agreement in the machine vision literature that objects and object categories should be represented as collections of *features* or *parts* with distinctive appearance and mutual position [1, 2, 4, 5, 6, 7, 8, 9]. A number of ideas for efficient detection algorithms (find instances of a given object category, e.g. faces) have been proposed by virtually all the cited authors, far fewer for recognition (list all objects and their pose in a given image) where matching would ideally take a logarithmic time with respect to the number of available models [3, 4]. Learning of parameters characterizing features shape or appearance is still a difficult area, with most authors opting for heavy human intervention (typically segmentation and alignment of the training examples, although [1, 2, 3] train without supervision) and very large training sets for object categories (typically in the order of $10^3$ - $10^4$, although [10] recently demonstrated learning categories from 1-10 examples).

This work is based on two complementary efforts: the deterministic recognition system proposed by Lowe [3, 4], and the probabilistic constellation models by Perona and collaborators [1, 2]. The first line of work has three attractive characteristics: objects are represented with hundreds of features, thus increasing robustness; models are learned from a single training example; last but not least, recognition is efficient with databases of hundreds of objects. The drawback of Lowe's approach is that both modeling decisions and algorithms rely on heuristics, whose design and performance may be far from optimal in

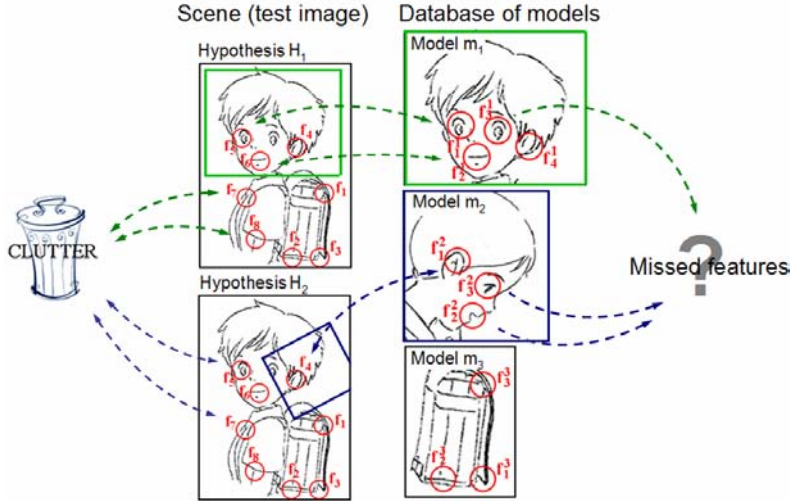

Figure 1: Diagram of our recognition model showing database, test image and two competing hypotheses. To avoid a cluttered diagram, only one partial hypothesis is displayed for each hypothesis. The predicted position of models according to the hypotheses are overlaid on the test image.

some circumstances. Conversely, the second line of work is based on principled probabilistic object models which yield principled and, in some respects, optimal algorithms for learning and recognition/detection. Unfortunately, the large number of parameters employed in each model limit in practice the number of features being used and require many training examples. By recasting Lowe's model and algorithms in probabilistic terms, we hope to combine the advantages of both methods. Besides, in this paper we choose to focus on individual objects as in [3, 4] rather than on categories as in [1, 2].

In [11] we presented a model aimed at the same problem of individual object recognition. A major difference with the work described here lies in the probabilistic treatment of hypotheses, which allows us here to use directly hypothesis likelihood as a guide for the search, instead of the arbitrary admissible heuristic required by A*.

## 2   Probabilistic framework and notations

Each model object is represented as a collection of *features*. Features are informative parts extracted from images by an interest point operator. Each model is the set of features extracted from one training image of a given object - although this could be generalized to features from many images of the same object. Models are indexed by $k$ and denoted by $m_k$, while indices $i$ and $j$ are used respectively for features extracted from the test image and from model images: $f_i$ denotes the $i - th$ test feature, while $f_j^k$ denotes the $j - th$ feature from the $k - th$ model. The features extracted from model images (training set) form the *database*. A feature detected in a test image can be a consequence of the presence of a model object in the image, in which case it should be associated to a feature from the database. In the alternative, this feature is attributed to a clutter - or background - detection.

The geometric information associated to each feature contains position information ($x$ and $y$ coordinates, denoted by the vector $\overline{x}$), orientation (denoted by $\theta$) and scale (denoted by $\sigma$). It is denoted by $\mathcal{X}_i = (\overline{x}, \theta_i, \sigma_i)$ for test feature $f_i$ and $\mathcal{X}_j^k = (\overline{x}_j^k \theta_j^k, \sigma_j^k)$ for model feature $f_j^k$. This geometric information is measured relatively to the standard *reference frame* of the image in which the feature has been detected. All features extracted from the same image share the same reference frame.

The appearance information associated to a feature is a descriptor characterizing the local image appearance near this feature. The measured appearance information is denoted by

$\mathcal{A}_i$ for test feature $f_i$ and $\mathcal{A}_j^k$ for model feature $f_j^k$. In our experiments, features are detected at multiple scales at the extrema of difference-of-gaussians filtered versions of the image [4, 12]. The SIFT descriptor [4] is then used to characterize the local texture about keypoints.

A *partial hypothesis* $h$ explains the observations made in a fraction of the test image. It combines a model image $m_h$ and a corresponding set of *pose parameters* $\mathcal{X}_h$. $\mathcal{X}_h$ encodes position, rotation, scale (this can easily be extended to affine transformations). We assume independence between partial hypotheses. This requires in particular independence between models. Although reasonable, this approximation is not always true (e.g. a keyboard is likely to be detected close to a computer screen). This allows us to search in parallel for multiple objects in a test image.

A *hypothesis H* is the combination of several partial hypotheses, such that it explains completely the observations made in the test image. A special notation $H_0$ or $h_0$ denotes any (partial) hypothesis that states that no model object is present in a given fraction of the test image, and that features that could have been detected there are due to clutter.

Our objective is to find which model objects are present in the test scene, given the observations made in the test scene and the information that is present in the database. In probabilistic terms, we look for hypotheses $H$ for which the likelihood ration $LR(H) = \frac{P(H|\{f_i\},\{f_j^k\})}{P(H_0|\{f_i\},\{f_j^k\})} > 1$. This ratio characterizes how well models and poses specified by $H$ explain the observations, as opposed to them being generated by clutter. Using Bayes rules and after simplifications,

$$LR(H) = \frac{P(H|\{f_i\},\{f_j^k\})}{P(H_0|\{f_i\},\{f_j^k\})} = \frac{P(\{f_i\}|\{f_j^k\},H) \cdot P(H)}{P(\{f_i\}|\{f_j^k\},H_0) \cdot P(H_0)} \qquad (1)$$

where we used $P(\{f_j^k\}|H) = P(\{f_j^k\})$ since the database observations do not depend on the current hypothesis.

A key assumption of this work is that once the pose parameters of the objects (and thus their reference frames) are known, the geometric configuration and appearance of the test features are independent from each other. We also assume independence between features associated to models and features associated to clutter detections, as well as independence between separate clutter detections. Therefore, $P(\{f_i\}|\{f_j^k\},H) = \prod_i P(f_i|\{f_j^k\},H)$. These assumptions of independence are also made in [13], and undelying in [4].

*Assignment vectors* $v$ represent matches between features from the test scene, and model features or clutter. The dimension of each assignment vector is the number of test features $n_{test}$. Its $i-th$ component $v(i) = (k,j)$ denotes that the test feature $f_i$ is matched to $f_{v(i)} = f_j^k$, $j-th$ feature from model $m_k$. $v(i) = (0,0)$ denotes the case where $f_i$ is attributed to clutter. The set $\mathcal{V}_H$ of assignment vectors compatible with a hypothesis $H$ are those that assign test features only to models present in $H$ (and to clutter). In particular, the only assignment vector compatible with $h_0$ is $v_0$ such that $\forall i, v_0(i) = (0,0)$. We obtain

$$LR(H) = \frac{P(H)}{P(H_0)} \sum_{v \in \mathcal{V}_H} \prod_{h \in H} \left( P(v|\{f_j^k\},m_h,\mathcal{X}_h) \cdot \prod_{i|f_i \in h} \frac{P(f_i|f_{v(i)},m_h,\mathcal{X}_h)}{P(f_i|h_0)} \right) \qquad (2)$$

$P(H)$ is a prior on hypotheses, we assume it is constant. The term $P(v|\{f_j^k\},m_h,\mathcal{X}_h)$ is discussed in 3.1, we now explore the other terms.

•$P(f_i|f_{v(i)},m_h,\mathcal{X}_h)$ : $f_i$ and $f_{v(i)}$ are believed to be one and the same feature. Differences measured between them are noise due to the imaging system as well as distortions caused by viewpoint or lighting conditions changes. This noise probability $p_n$ encodes differences in appearance of the descriptors, but also in geometry, i.e. position, scale, orientation Assuming independence between appearance information and geometry information,

$$p_n(f_i|f_j^k,m_h,\mathcal{X}_h) = p_{n,\mathcal{A}}(\mathcal{A}_i|\mathcal{A}_{v(i)},m_h,\mathcal{X}_h) \cdot p_{n,\mathcal{X}}(\mathcal{X}_i|\mathcal{X}_{v(i)},m_h,\mathcal{X}_h) \qquad (3)$$

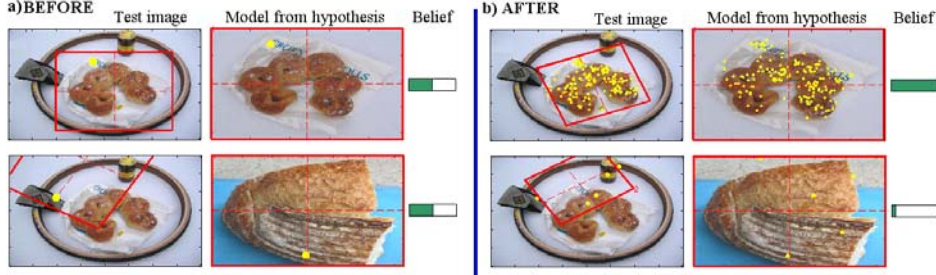

Figure 2: Snapshots from the iterative matching process. Two competing hypotheses are displayed (top and bottom row) a) Each assignment vector contains one assignment, suggesting a transformation (red box) b) End of iterative process. The correct hypothesis is supported by numerous matches and high belief, while the wrong hypothesis has only a weak support from few matches and low belief.

The error in geometry is measured by comparing the values observed in the test image, with the predicted values that would be observed if the model features were to be transformed according to the parameters $\mathcal{X}_h$. Let's denote by $\mathcal{X}_h(\overline{x}_{v(i)}), \mathcal{X}_h(\sigma_{v(i)}), \mathcal{X}_h(\theta_{v(i)})$ those predicted values, the geometry part of the noise probability can be decomposed into

$$p_{n,\mathcal{X}}(\mathcal{X}_i|\mathcal{X}_{v(i)}, h) = p_{n,\overline{x}}(\overline{x}_i, \mathcal{X}_h(\overline{x}_{v(i)})) \cdot p_{n,\sigma}(\sigma_i, \mathcal{X}_h(\sigma_{v(i)})) \cdot p_{n,\theta}(\theta_i, \mathcal{X}_h(\theta_{v(i)})) \quad (4)$$

• $P(f_i|h_0)$ is a density on appearance and position of clutter detections, denoted by $p_{bg}(f_i)$. We can decompose this density as well into an appearance term and a geometry term:

$$p_{bg}(f_i) = p_{bg,\mathcal{A}}(\mathcal{A}_i) \cdot p_{bg,\mathcal{X}}(\mathcal{X}_i) = p_{bg,\mathcal{A}}(\mathcal{A}_i) \cdot p_{bg,(\overline{x})}(\overline{x}_i) \cdot p_{bg,\sigma}(\sigma_i) \cdot p_{bg,\theta}(\theta_i) \quad (5)$$

$p_{bg,\mathcal{A}}$, $p_{bg,(\overline{x})}(\overline{x}_i)$ $p_{bg,\sigma}(\sigma_i)$, $p_{bg,\theta}(\theta_i)$ are densities that characterize, for clutter detections, appearance, position, scale and rotation respectively.

Out of lack of space, and since it is not the main focus of this paper, we will not go into the details of how the "foreground density" $p_n$ and the "background density" $p_{bg}$ are learned. The main assumption is that those densities are shared across features, instead of having one set of parameters for each feature as in [1, 2]. This results in an important decrease of the number of parameters to be learned, at a slight cost in the model expressiveness.

## 3   Search for the best interpretation of the test image

The building block of the recognition process is a *question*, comparing a feature from a database model with a feature of the test image. A question selects a feature from the database, and tries to identify if and where this feature appears in the test image.

### 3.1   Assignment vectors compatible with hypotheses

For a given hypothesis $H$, the set of possible assignment vectors $\mathcal{V}_H$ is too large for explicit exploration. Indeed, each potential match can either be accepted or rejected, which creates a combinatorial explosion. Hence, we approximate the summation in (2) by its largest term. In particular, each assignment vector $v$ and each model referenced in $v$ implies a set of pose parameters $\mathcal{X}_v$ (extracted e.g. with least-squares fitting). Therefore, the term $P(v|\{f_j^k\}, m_h, \mathcal{X}_h)$ from (2) will be significant only when $\mathcal{X}_v \approx \mathcal{X}_h$, i.e. when the pose implied by the assignment vector agrees with the pose specified by the partial hypothesis. We consider only the assignment vectors $v$ for which $\mathcal{X}_v \approx \mathcal{X}_h$. $P(v_H|\{f_j^k\}, h)$ is assumed to be close to 1. Eq.(2) becomes

$$LR(H) \approx \frac{P(H)}{P(H_0)} \prod_{h \in H} \prod_{i|f_i \in h} \frac{P(f_i|f_{v_h(i)}, m_h, \mathcal{X}_h)}{P(f_i|h_0)} \quad (6)$$

Our recognition system proceeds by asking questions sequentially and adding matches to assignment vectors. It is therefore natural to define, for a given hypothesis $H$ and the corresponding assignment vector $v_H$ and $t \leq n_{test}$, the *belief in $v_H$* by

$$B_0(v_H) = 1, \quad B_t(v_H) = \frac{p_n(f_t|f_{v(t)}, m_{h_t}, \mathcal{X}_{h_t})}{p_{bg}(f_t|h_0)} \cdot B_{t-1}(v_H) \tag{7}$$

The geometric part of the belief (cf.(3)-(5) characterizes how close the pose $\mathcal{X}_v$ implied by the assignments is to the pose $\mathcal{X}_h$ specified by the hypothesis. The geometric component of the belief characterizes the quality of the appearance match for the pairs $(f_i, f_{v(i)})$.

## 3.2 Entropy-based optimization

Our goal is finding quickly the hypothesis that best explains the observations, i.e. the hypothesis (models+poses) that has the highest likelihood ratio. We compute such hypothesis incrementally by asking questions sequentially. Each time a question is asked we update the beliefs. We stop the process and declare a detection (i.e. a given model is present in the image) as soon as the belief of a corresponding hypothesis exceeds a given confidence threshold. The speed with which we reach such a conclusion depends on choosing cleverly the next question. A greedy strategy says that the best next question is the one that takes us closest to a detection decision. We do so by considering the entropy of the vector of beliefs (the vector may be normalized to 1 so that each belief is in fact a probability): the lower the entropy the closer we are to a detection. Therefore we study the following heuristic: *The most informative next question is the one that minimizes the expectation of the entropy of our beliefs*. We call this strategy 'minimum expected entropy' (MEE). This idea is due to Geman et al. [14].

Calculating the MEE question is, unfortunately, a complex and expensive calculation in itself. In Monte-Carlo simulations of a simplified version of our problem we notice that the MEE strategy tends to ask questions that relate to the maximum-belief hypothesis. Therefore we approximate the MEE strategy with a simple heuristic: *The next question consists of attempting to match one feature of the highest-belief model; specifically, the feature with best appearance match to a feature in the test image.*

## 3.3 Search for the best hypotheses

In an initialization step, a geometric hash table [3, 6, 7] is created by discretizing the space of possible transformations Note that we add only partial hypotheses in a hypothesis one at a time, which allows us to discretize only the space of partial hypotheses (models + poses), instead of discretizing the space of *combinations* of partial hypotheses.

Questions to be examined are created by pairing database features to the test features closest in terms of appearance. Note that since features encode location, orientation and scale, any single assignment between a test feature and a model feature contains enough information to characterize a similarity transformation. It is therefore natural to restrict the set of possible transformations to similarities, and to insert each candidate assignment in the corresponding geometric hash table entry. This forms a pool of candidate assignments. The set of hypotheses is initialized to the center of the hash table entries, and their belief is set to 1. The motivation for this initialization step is to examine, for each partial hypothesis, only a small number of candidate matches. A partial hypothesis corresponds to a hash table entry, we consider only the candidate assignments that fall into this same entry.

Each iteration proceeds as follows. The hypothesis $H$ that currently has the highest likelihood ratio is selected. If the geometric hash table entry corresponding to the current partial hypothesis $h$, contains candidate assignments that have not been examined yet, one of them, $(f_i, f_j^{m_h})$ is picked - currently, the best appearance match - and the probabilities $p_{bg}(f_i)$ and $p_n(f_i|f_j^{m_h}, m_h, \mathcal{X}_h)$ are computed. As mentioned in 3.1, only the best assignment

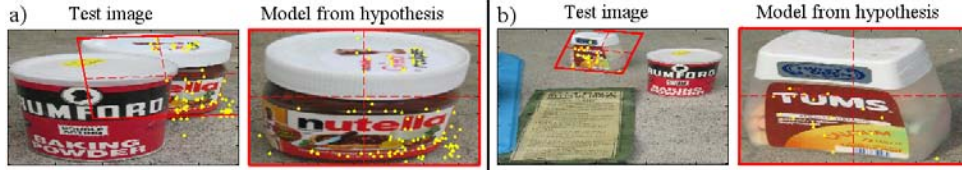

Figure 3: Results from our algorithm in various situations (viewpoint change can be seen in Fig.6). Each row shows the best hypothesis in terms of belief. a) Occlusion b) Change of scale.

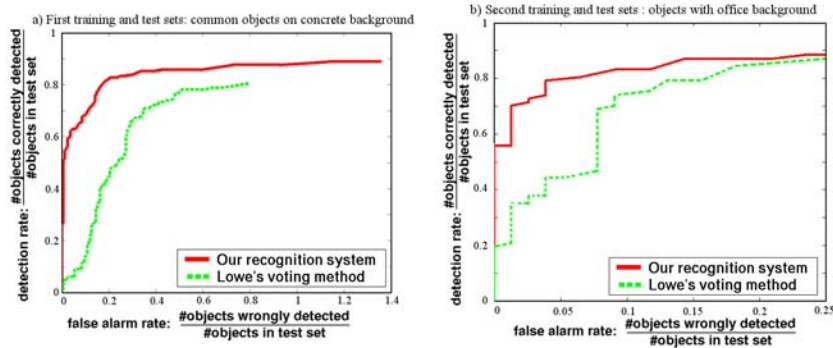

Figure 4: ROC curves for both experiments. The performance improvement from our probabilistic formulation is particularly significant when a low false alarm rate is desired. The threshold used is the repeatability rate defined in [15]

vector is explored: if $p_n(f_i|f_j^{m_h}, m_h, \mathcal{X}_h) > p_{bg}(f_i)$ the match is accepted and inserted in the hypothesis. In the alternative, $f_i$ is considered a clutter detection and $f_j^{m_h}$ is a missed detection. The belief $B(v_H)$ and the likelihood ratio $LR(H)$ are updated using (7).

After adding an assignment to a hypothesis, frame parameters $\mathcal{X}_h$ are recomputed using least-squares optimization, based on all assignments currently associated to this hypothesis. This parameter estimation step provides a progressive refinement of the model pose parameters as assignments are added. Fig.2 illustrates this process.

The exploration of a partial hypothesis ends when no more candidate match is available in the hash table entry. We proceed with the next best partial hypothesis. The search ends when all test scene features have been matched or assigned to clutter.

## 4 Experimental results

### 4.1 Experimental setting

We tested our algorithm on two sets of images, containing respectively 49 and 161 model images, and 101 and 51 test images (sets $PM - gadgets - 03$ and $JP - 3Dobjects - 04$ available from $http : //www.vision.caltech.edu/html - files/archive.html$). Each model image contained a single object. Test images contained from zero (negative examples) to five objects, for a total of 178 objects in the first set, and 79 objects in the second set. A large fraction of each test image consists of background. The images were taken with no precautions relatively to lighting conditions or viewing angle.

The first set contains common kitchen items and objects of everyday use. The second set (Ponce Lab, UIUC) includes office pictures. The objects were always moved between model images and test images. The images of model objects used in the learning stage were downsampled to fit in a $500 \times 500$ pixels box, the test images were downsampled to $800 \times 800$ pixels. With these settings, the number of features generated by the features detector was of the order of 1000 per training image and 2000-4000 per test image.

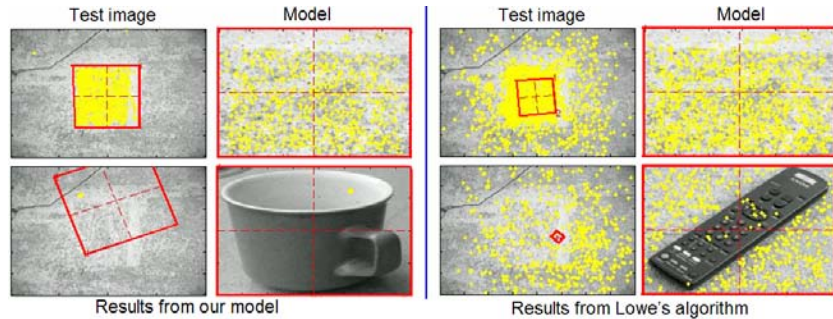

Figure 5: Behavior induced by clutter detections. A ground truth model was created by cutting a rectangle from the test image and adding noise. The recognition process is therefore expected to find a perfect match. The two rows show the best and second best model found by each algorithm (estimated frame position shown by the red box, features that found a match are shown in yellow).

## 4.2 Results

Our probabilistic method was compared against Lowe's voting approach on both sets of images. We implemented Lowe's algorithm following the details provided in [3, 4]. Direct comparison of our approach to 'constellation' models [1, 2] is not possible as those require many training samples for each class in order to learn shape parameters, while our method learns from single examples. Recognition time for our unoptimized implementations was 10 seconds for Lowe's algorithm and 25 seconds for our probabilistic method on a 2.8GHz PC, both implementations used approximately 200MB of memory.

Both methods yielded similar detection rates for simple scenes. In challenging situations with multiple objects or textured clutter, our method performs a more systematic check on geometric consistency by updating likelihoods every time a match is added. Hypotheses starting with wrong matches due to clutter don't find further supporting matches, and are easily discarded by a threshold based on the number of matches. Conversely, Lowe's algorithm checks geometric consistency as a last step of the recognition process, but needs to allow for a large slop in the transformation parameters. Spurious matches induced by clutter detections may still be accepted, thus leading to the acceptance of incorrect hypotheses.

An example of this behavior is displayed in Fig.5: the test image consists of a picture of concrete. A rectangular patch was extracted from this image, noise was added to this patch, and it was inserted in the database as a new model. With our algorithm, the best hypothesis found the correct match with the patch of concrete, its best contender doesn't succeed in collecting more than one correspondence and is discarded. In Lowe's case, other models manage to accumulate a high number of correspondences induced by texture matches among clutter detections. Although the first correspondence concerns the correct model, it contains wrong matches. Moreover, the model displayed in the second row leads to a false alarm supported by many matches.

Fig.4 displays receiver-operating curves (ROC) for both tests sets, obtained for our probabilistic system and Lowe's method. Both curves confirm that our probabilistic interpretation leads to less false alarms than Lowe's method for a same detection rate.

## 5 Conclusion

We have proposed an object recognition method that combines the benefits of a set of rich features with those of a probabilistic model of features positions and appearance. The use of large number of features brings robustness with respect to occlusions and clutter. The probabilistic model verifies the validity of candidate hypotheses in terms of appearance and geometric configuration. Our system improves upon a state-of-the art recognition method based on strict feature matching. In particular, the rate of false alarms in the presence

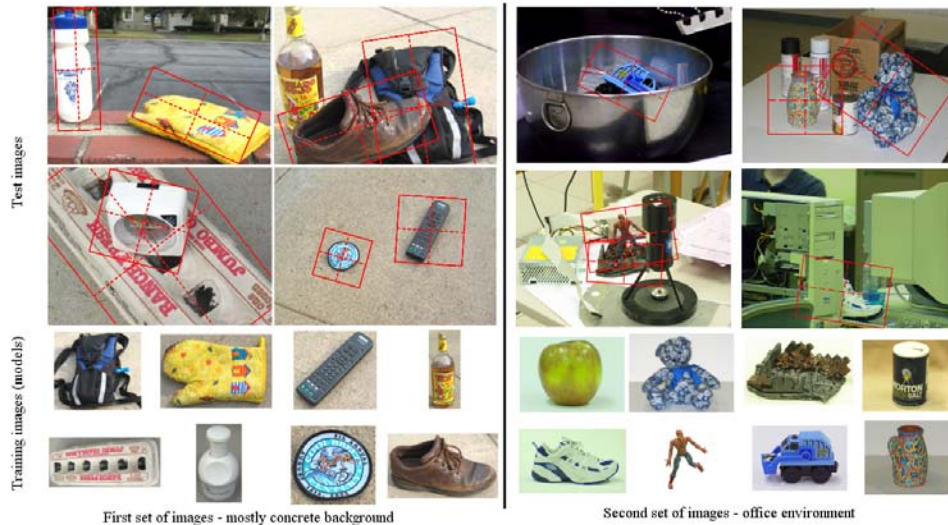

Figure 6: Sample scenes and training objects from the two sets of images. Recognized frame poses are overlayed in red.

of textured backgrounds generating strong erroneous matches, is lower. This is a strong advantage in real-world situations, where a "clean" background is not always available.

## References

[1] M. Weber, M. Welling and P. Perona, "Unsupervised Learning of Models for Recognition", *Proc. Europ. Conf. Comp. Vis.*, 2000.

[2] R. Fergus, P. Perona, A. Zisserman, "Object Class Recognition by Unsupervised Scale-invariant Learning", *IEEE. Conf. on Comp. Vis. and Patt. Recog.*, 2003.

[3] D.G. Lowe, "Object Recognition from Local Scale-invariant Features", *ICCV*,1999

[4] D.G. Lowe, "Distinctive Image Features from Scale-Invariant Keypoints", *Int. J. Comp. Vis.*, 60(2), pp. 91-110, 2004.

[5] G. Carneiro and A. Jepson "Flexible Spatial Models for Grouping Local Image Features", *IEEE. Conf. on Comp. Vis. and Patt. Recog.*, 2004.

[6] I. Rigoutsos and R. Hummel "A Bayesian Approach to Model Matching with Geometric Hashing", *CVIU*, 62(1), pp. 11-26, 1995.

[7] W.E.L. Grimson and D.P. Huttenlocher, "On the Sensitivity of Geometric Hashing", *ICCV*, 1990

[8] H. Rowley, S. Baluja, T. Kanade, "Neural Network-based Face Detection", *IEEE. Trans. Patt. Anal. Mach. Int.*, 20(1):pp. 23-38, 1998.

[9] P. Viola and M. Jones, "Rapid Object Detection Using a Boosted Cascade of Simple Features", *Proc. IEEE Conf. Comp. Vis. Patt. Recog.*, 2001.

[10] L. Fei-Fei, R. Fergus, P. Perona. "Learning Generative Visual Models from Few Training Examples: An Incremental Bayesian Approach Tested on 101 Object Categories" *CVPR*, 2004.

[11] P. Moreels, M. Maire, P. Perona, 'Recognition by Probabilistic Hypothesis Construction', *Proc. 8th Europ. Conf. Comp. Vision*, Prague, Czech Republic, pp.55-68, 2004

[12] T. Lindeberg, "Scale-space Theory: a Basic Tool for Analising Structures at Different Scales", *J. Appl. Stat.*, 21(2), pp.225-270, 1994.

[13] A.R. Pope and D.G. Lowe, "Probabilistic Models of Appearance for 3-D Object Recognition", *Int. J. Comp. Vis.*, 40(2), pp. 149-167, 2000.

[14] D. Geman and B. Jedynak, "An Active Testing Model for Tracking Roads in Satellite Images", *IEEE. Trans. Patt. Anal. Mach. Int.*,18(1) pp. 1 - 14,1996

[15] C. Schmid, R. Mohr, C. Bauckhage", "Comparing and Evaluating Interest Points", *Proc. of 6th Int. Conf. Comp. Vis.*, Bombay, India, 1998.
